# Asymptotic Theory for Regularization: One-Dimensional Linear Case

**Petri Koistinen**
Rolf Nevanlinna Institute, P.O. Box 4, FIN-00014 University of Helsinki,
Finland. Email: Petri.Koistinen@rni.helsinki.fi

## Abstract

The generalization ability of a neural network can sometimes be improved dramatically by regularization. To analyze the improvement one needs more refined results than the asymptotic distribution of the weight vector. Here we study the simple case of one-dimensional linear regression under quadratic regularization, i.e., ridge regression. We study the random design, misspecified case, where we derive expansions for the optimal regularization parameter and the ensuing improvement. It is possible to construct examples where it is best to use no regularization.

## 1 INTRODUCTION

Suppose that we have available training data $(X_1, Y_1), \ldots, (X_n, Y_n)$ consisting of pairs of vectors, and we try to predict $Y_i$ on the basis of $X_i$ with a neural network with weight vector $w$. One popular way of selecting $w$ is by the criterion

$$(1) \qquad \frac{1}{n} \sum_1^n \ell(X_i, Y_i, w) + \lambda Q(w) = \min!,$$

where the loss $\ell(x, y, w)$ is, e.g., the squared error $\|y - g(x, w)\|^2$, the function $g(\cdot, w)$ is the input/output function of the neural network, the penalty $Q(w)$ is a real function which takes on small values when the mapping $g(\cdot, w)$ is smooth and high values when it changes rapidly, and the regularization parameter $\lambda$ is a nonnegative scalar (which might depend on the training sample). We refer to the setup (1) as (training with) regularization, and to the same setup with the choice $\lambda = 0$ as training without regularization. Regularization has been found to be very effective for improving the generalization ability of a neural network especially when the sample size $n$ is of the same order of magnitude as the dimensionality of the parameter vector $w$, see, e.g., the textbooks (Bishop, 1995; Ripley, 1996).

In this paper we deal with asymptotics in the case where the architecture of the network is fixed but the sample size grows. To fix ideas, let us assume that the training data is part of an i.i.d. (independent, identically distributed) sequence $(X, Y); (X_1, Y_1), (X_2, Y_2), \ldots$ of pairs of random vectors, i.e., for each $i$ the pair $(X_i, Y_i)$ has the same distribution as the pair $(X, Y)$ and the collection of pairs is independent ($X$ and $Y$ can be dependent). Then we can define the (prediction) risk of a network with weights $w$ as the expected value

$$(2) \qquad r(w) := \mathbb{E}\,\ell(X, Y, w).$$

Let us denote the minimizer of (1) by $\hat{w}_n(\lambda)$, and a minimizer of the risk $r$ by $w^*$. The quantity $r(\hat{w}_n(\lambda))$ is the average prediction error for data independent of the training sample. This quantity $r(\hat{w}_n(\lambda))$ is a random variable which describes the generalization performance of the network: it is bounded below by $r(w^*)$ and the more concentrated it is about $r(w^*)$, the better the performance. We will quantify this concentration by a single number, the expected value $\mathbb{E}\,r(\hat{w}_n(\lambda))$. We are interested in quantifying the gain (if any) in generalization for training with versus training without regularization defined by

$$(3) \qquad \mathbb{E}\,r(\hat{w}_n(0)) - \mathbb{E}\,r(\hat{w}_n(\lambda)).$$

When regularization helps, this is positive.

However, relatively little can be said about the quantity (3) without specifying in detail how the regularization parameter is determined. We show in the next section that provided $\lambda$ converges to zero sufficiently quickly (at the rate $o_p(n^{-1/2})$), then $\mathbb{E}\,r(\hat{w}_n(0))$ and $\mathbb{E}\,r(\hat{w}_n(\lambda))$ are equal to leading order. It turns out, that the optimal regularization parameter resides in this asymptotic regime. For this reason, delicate analysis is required in order to get an asymptotic approximation for (3). In this article we derive the needed asymptotic expansions only for the simplest possible case: one-dimensional linear regression where the regularization parameter is chosen independently of the training sample.

## 2 REGULARIZATION IN LINEAR REGRESSION

We now specialize the setup (1) to the case of linear regression and a quadratic smoothness penalty, i.e., we take $\ell(x, y, w) = [y - x^T w]^2$ and $Q(w) = w^T R w$, where now $y$ is scalar, $x$ and $w$ are vectors, and $R$ is a symmetric, positive definite matrix. It is well known (and easy to show) that then the minimizer of (1) is

$$(4) \qquad \hat{w}_n(\lambda) = \left[\frac{1}{n}\sum_1^n X_i X_i^T + \lambda R\right]^{-1} \frac{1}{n}\sum_1^n X_i Y_i.$$

This is called the *generalized ridge regression estimator*, see, e.g., (Titterington, 1985); ridge regression corresponds to the choice $R = I$, see (Hoerl and Kennard, 1988) for a survey. Notice that (generalized) ridge regression is usually studied in the *fixed design* case, where $X_i$:s are nonrandom. Further, it is usually assumed that the model is *correctly specified*, i.e., that there exists a parameter such that $Y_i = X_i^T w^* + \epsilon_i$, and such that the distribution of the noise term $\epsilon_i$ does not depend on $X_i$. In contrast, we study the *random design, misspecified* case.

Assuming that $\mathbb{E}\,\|X\|^2 < \infty$ and that $\mathbb{E}[XX^T]$ is invertible, the minimizer of the risk (2) and the risk itself can be written as

$$(5) \qquad w^* = A^{-1}\mathbb{E}[XY], \quad \text{with} \quad A := \mathbb{E}[XX^T]$$

$$(6) \qquad r(w) = r(w^*) + (w - w^*)^T A(w - w^*).$$

If $Z_n$ is a sequence of random variables, then the notation $Z_n = o_p(n^{-\alpha})$ means that $n^\alpha Z_n$ converges to zero in probability as $n \to \infty$. For this notation and the mathematical tools needed for the following proposition see, e.g., (Serfling, 1980, Ch. 1) or (Brockwell and Davis, 1987, Ch. 6).

**Proposition 1** *Suppose that* $\mathbb{E} Y^4 < \infty$, $\mathbb{E} \|X\|^4 < \infty$ *and that* $A = \mathbb{E}[XX^T]$ *is invertible. If* $\lambda = o_p(n^{-1/2})$, *then both* $\sqrt{n}(\hat{w}_n(0) - w^*)$ *and* $\sqrt{n}(\hat{w}_n(\lambda) - w^*)$ *converge in distribution to* $N(0, C)$, *a normal distribution with mean zero and covariance matrix* $C$.

The previous proposition also generalizes to the nonlinear case (under more complicated conditions). Given this proposition, it follows (under certain additional conditions) by Taylor expansion that both $\mathbb{E} r(\hat{w}_n(\lambda)) - r(w^*)$ and $\mathbb{E} r(\hat{w}_n(0)) - r(w^*)$ admit the expansion $\beta_1 n^{-1} + o(n^{-1})$ with the same constant $\beta_1$. Hence, in the regime $\lambda = o_p(n^{-1/2})$ we need to consider higher order expansions in order to compare the performance of $\hat{w}_n(\lambda)$ and $\hat{w}_n(0)$.

## 3  ONE-DIMENSIONAL LINEAR REGRESSION

We now specialize the setting of the previous section to the case where $x$ is scalar. Also, from now on, we only consider the case where the regularization parameter for given sample size $n$ is deterministic; especially $\lambda$ is not allowed to depend on the training sample. This is necessary, since coefficients in the following type of asymptotic expansions depend on the details of how the regularization parameter is determined. The deterministic case is the easiest one to analyze.

We develop asymptotic expansions for the criterion

$$(7) \qquad J_n(k) := \mathbb{E}\left(r(\hat{w}_n(k))\right) - r(w^*),$$

where now the regularization parameter $k$ is deterministic and nonnegative. The expansions we get turn out to be valid uniformly for $k \geq 0$. We then develop asymptotic formulas for the minimizer of $J_n$, and also for $\bar{J}_n(0) - \inf J_n$. The last quantity can be interpreted as the average improvement in generalization performance gained by optimal level of regularization, when the regularization constant is allowed to depend on $n$ but not on the training sample.

From now on we take $Q(w) = w^2$ and assume that $A = \mathbb{E} X^2 = 1$ (which could be arranged by a linear change of variables). Referring back to formulas in the previous section, we see that

$$(8) \qquad r(\hat{w}_n(k)) - r(w^*) = (\bar{V}_n - kw^*)^2/(\bar{U}_n + 1 + k)^2 =: h(\bar{U}_n, \bar{V}_n, k),$$

whence $J_n(k) = \mathbb{E} h(\bar{U}_n, \bar{V}_n, k)$, where we have introduced the function $h$ (used heavily in what follows) as well as the arithmetic means $\bar{U}_n$ and $\bar{V}_n$

$$(9) \qquad \bar{U}_n := \frac{1}{n}\sum_1^n U_i, \quad \bar{V}_n := \frac{1}{n}\sum_1^n V_i, \quad \text{with}$$

$$(10) \qquad U_i := X_i^2 - 1, \quad V_i := X_i Y_i - w^* X_i^2$$

For convenience, also define $U := X^2 - 1$ and $V := XY - w^* X^2$. Notice that $U; U_1, U_2, \ldots$ are zero mean i.i.d. random variables, and that $V; V_1, V_2, \ldots$ satisfy the same conditions. Hence $\bar{U}_n$ and $\bar{V}_n$ converge to zero, and this leads to the idea of using the Taylor expansion of $h(u, v, k)$ about the point $(u, v) = (0, 0)$ in order to get an expansion for $J_n(k)$.

To outline the ideas, let $T_j(u,v,k)$ be the degree $j$ Taylor polynomial of $(u,v) \mapsto h(u,v,k)$ about $(0,0)$, i.e., $T_j(u,v,k)$ is a polynomial in $u$ and $v$ whose coefficients are functions of $k$ and whose degree with respect to $u$ and $v$ is $j$. Then $\mathbb{E} T_j(\bar{U}_n, \bar{V}_n, k)$ depends on $n$ and moments of $U$ and $V$. By deriving an upper bound for the quantity $\mathbb{E}|h(\bar{U}_n, \bar{V}_n, k) - T_j(\bar{U}_n, \bar{V}_n, k)|$ we get an upper bound for the error committed in approximating $J_n(k)$ by $\mathbb{E} T_j(\bar{U}_n, \bar{V}_n, k)$. It turns out that for odd degrees $j$ the error is of the same order of magnitude in $n$ as for degree $j-1$. Therefore we only consider even degrees $j$. It also turns out that the error bounds are uniform in $k \geq 0$ whenever $j \geq 2$. To proceed, we need to introduce assumptions.

**Assumption 1** $\mathbb{E}|X|^r < \infty$ and $\mathbb{E}|Y|^s < \infty$ for high enough $r$ and $s$.

**Assumption 2** *Either (a) for some constant $\beta > 0$ almost surely $|X| \geq \beta$ or (b) $X$ has a density which is bounded in some neighborhood of zero.*

Assumption 1 guarantees the existence of high enough moments; the values $r = 20$ and $s = 8$ are sufficient for the following proofs. E.g., if the pair $(X,Y)$ has a normal distribution or a distribution with compact support, then moments of all orders exist and hence in this case assumption 1 would be satisfied. Without some condition such as assumption 2, $J_n(0)$ might fail to be meaningful or finite. The following technical result is stated without proof.

**Proposition 2** *Let $p > 0$ and let $0 < \mathbb{E} X^2 < \infty$. If assumption 2 holds, then*

$$\mathbb{E}\left\{\left[\frac{1}{n}(X_1^2 + \cdots + X_n^2)\right]^{-p}\right\} \to \left[\mathbb{E}(X^2)\right]^{-p}, \quad as \quad n \to \infty,$$

*where the expectation on the left is finite (a) for $n \geq 1$ (b) for $n > 2p$ provided that assumption 2 (a), respectively 2 (b) holds.*

**Proposition 3** *Let assumptions 1 and 2 hold. Then there exist constants $n_0$ and $M$ such that*

$$J_n(k) = \mathbb{E} T_2(\bar{U}_n, \bar{V}_n, k) + R(n,k) \quad where$$

$$\mathbb{E} T_2(\bar{U}_n, \bar{V}_n, k) = \frac{(w^*)^2 k^2}{(1+k)^2} + n^{-1}\left[\frac{\mathbb{E} V^2}{(1+k)^2} + 3\frac{(w^*)^2 k^2 \mathbb{E} U^2}{(1+k)^4} + 4\frac{w^* k \mathbb{E} UV}{(1+k)^3}\right]$$

$$|R(n,k)| \leq Mn^{-3/2}(k+1)^{-1}, \quad \forall n \geq n_0, k \geq 0.$$

PROOF SKETCH The formula for $\mathbb{E} T_2(\bar{U}_n, \bar{V}_n, k)$ follows easily by integrating the degree two Taylor polynomial term by term. To get the upper bound for $R(n,k)$, consider the residual

$$h(u,v,k) - T_2(u,v,k) = \frac{-2(k+1)^3 uv^2 + -4(w^*)^2 k^2(k+1)u^3 + \cdots}{(u+1+k)^2(k+1)^4},$$

where we have omitted four similar terms. Using the bound

$$(\bar{U}_n + 1 + k)^2 = \left(\frac{1}{n}\sum_1^n X_i^2 + k\right)^2 \geq \left(\frac{1}{n}\sum_1^n X_i^2\right)^2, \quad \forall k \geq 0,$$

the $L_1$ triangle inequality, and the Cauchy-Schwartz inequality, we get

$$|R(n,k)| = |\mathbb{E}\left[h(\bar{U}_n, \bar{V}_n, k) - T_2(\bar{U}_n, \bar{V}_n, k)\right]|$$

$$\leq (k+1)^{-4} \left\{ \mathbb{E}\left[ (\frac{1}{n}\sum_1^n X_i^2)^{-4} \right] \right\}^{1/2}$$

$$\left\{ 2(k+1)^3 [\mathbb{E}\left(|\bar{U}_n|^2 |\bar{V}_n|^4\right)]^{1/2} + 4(w^*)^2 k^2 (k+1)[\mathbb{E}|\bar{U}_n|^6]^{1/2} \dots \right\}$$

By proposition 2, here $\mathbb{E}[(\frac{1}{n}\sum_1^n X_i^2)^{-4}] = O(1)$. Next we use the following fact, cf. (Serfling, 1980, Lemma B, p. 68).

**Fact 1** *Let $\{Z_i\}$ be i.i.d. with $\mathbb{E}[Z_1] = 0$ and with $\mathbb{E}|Z_1|^\nu < \infty$ for some $\nu \geq 2$. Then*

$$\mathbb{E}\left| \frac{1}{n}\sum_1^n Z_i \right|^\nu = O(n^{-\nu/2})$$

Applying the Cauchy-Schwartz inequality and this fact, we get, e.g., that

$$[\mathbb{E}\left(|\bar{U}_n|^2 |\bar{V}_n|^4\right)]^{1/2} \leq [(\mathbb{E}|\bar{U}_n|^4)^{1/2}(\mathbb{E}|\bar{V}_n|^8)^{1/2}]^{1/2} = O(n^{-3/2}).$$

Going through all the terms carefully, we see that the bound holds.                    □

**Proposition 4** *Let assumptions 1 and 2 hold, assume that $w^* \neq 0$, and set*

$$\alpha_1 := (\mathbb{E}\, V^2 - 2w^*\mathbb{E}[UV])/(w^*)^2.$$

*If $\alpha_1 > 0$, then there exists a constant $n_1$ such that for all $n \geq n_1$ the function $k \mapsto \mathbb{E}\, T_2(\bar{U}_n, \bar{V}_n, k)$ has a unique minimum on $[0, \infty)$ at the point $k_n^*$ admitting the expansion*

$$k_n^* = \alpha_1 n^{-1} + O(n^{-2}); \quad further,$$

$$J_n(0) - \inf\{J_n(k) : k \geq 0\} = J_n(0) - J_n(\alpha_1 n^{-1}) = \alpha_1^2(w^*)^2 n^{-2} + O(n^{-5/2}).$$

*If $\alpha \leq 0$, then*

$$\inf\{J_n(k) : k \geq 0\} = J_n(0) + O(n^{-5/2}).$$

PROOF SKETCH The proof is based on perturbation expansion considering $1/n$ a small parameter. By the previous proposition, $S_n(k) := \mathbb{E}\, T_2(\bar{U}_n, \bar{V}_n, k)$ is the sum of $(w^*)^2 k^2/(1+k)^2$ and a term whose supremum over $k \geq k_0 > -1$ goes to zero as $n \to \infty$. Here the first term has a unique minimum on $(-1, \infty)$ at $k = 0$. Differentiating $S_n$ we get

$$S_n'(k) = [2(w^*)^2 k(k+1)^2 + n^{-1}p_2(k)]/(k+1)^5,$$

where $p_2(k)$ is a second degree polynomial in $k$. The numerator polynomial has three roots, one of which converges to zero as $n \to \infty$. A regular perturbation expansion for this root, $k_n^* = \alpha_1 n^{-1} + \alpha_2 n^{-2} + \dots$, yields the stated formula for $\alpha_1$. This point is a minimum for all sufficiently large $n$; further, it is greater than zero for all sufficiently large $n$ if and only if $\alpha_1 > 0$.

The estimate for $J_n(0) - \inf\{J_n(k) : k \geq 0\}$ in the case $\alpha_1 > 0$ follows by noticing that

$$J_n(0) - J_n(k) = \mathbb{E}\left[h(\bar{U}_n, \bar{V}_n, 0) - h(\bar{U}_n, \bar{V}_n, k)\right],$$

where we now use a third degree Taylor expansion about $(u, v, k) = (0, 0, 0)$

$$h(u, v, 0) - h(u, v, k) =$$

$$2w^* kv - (w^*)^2 k^2 - 4w^* kuv + 2(w^*)^2 k^2 u + 2kv^2 - 4w^* k^2 v + 2(w^*)^2 k^3 + r(u, v, k).$$

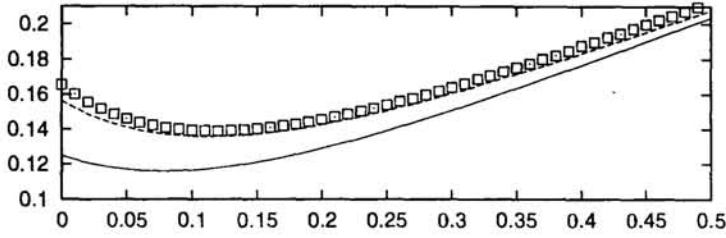

Figure 1: Illustration of the asymptotic approximations in the situation of equation (11). Horizontal axis $k$; vertical axis $J_n(k)$ and its asymptotic approximations. Legend: markers $J_n(k)$; solid line $\mathbb{E}\,T_2(\bar{U}_n, \bar{V}_n, k)$; dashed line $\mathbb{E}\,T_4(\bar{U}_n, \bar{V}_n, k)$.

Using the techniques of the previous proposition, it can be shown that $\mathbb{E}\,|r(\bar{U}_n, \bar{V}_n, k_n^*)| = O(n^{-5/2})$. Integrating the Taylor polynomial and using this estimate gives

$$J_n(0) - J_n(\alpha_1/n) = \alpha_1^2 (w^*)^2 n^{-2} + O(n^{-5/2}).$$

Finally, by the mean value theorem,

$$J_n(0) - \inf\{J_n(k) : k \geq 0\} = J_n(0) - J_n(\alpha_1/n) + \frac{d}{dk}[J_n(0) - J_n(k)]_{|k=\theta}(k_n^* - \alpha_1/n)$$

$$= J_n(0) - J_n(\alpha_1/n) + O(n^{-1})O(n^{-2})$$

where $\theta$ lies between $k_n^*$ and $\alpha_1/n$, and where we have used the fact that the indicated derivative evaluated at $\theta$ is of order $O(n^{-1})$, as can be shown with moderate effort. $\qquad\square$

**Remark** In the preceding we assumed that $A = \mathbb{E}\,X^2$ equals 1. If this is not the case, then the formula for $\alpha_1$ has to be divided by $A$; again, if $\alpha_1 > 0$, then $k_n^* = \alpha_1 n^{-1} + O(n^{-2})$.

If the model is correctly specified in the sense that $Y = w^* X + \epsilon$, where $\epsilon$ is independent of $X$ and $\mathbb{E}\,\epsilon = 0$, then $V = X\epsilon$ and $\mathbb{E}\,[UV] = 0$. Hence we have $\alpha_1 = \mathbb{E}\,[\epsilon^2]/(w^*)^2$, and this is strictly positive expect in the degenerate case where $\epsilon = 0$ with probability one. This means that here regularization helps provided the regularization parameter is chosen around the value $\alpha_1/n$ and $n$ is large enough. See Figure 1 for an illustration in the case

(11)
$$X \sim N(0,1), \quad Y = w^* X + \epsilon, \quad \epsilon \sim N(0,1), \quad w^* = 1,$$

where $\epsilon$ and $X$ are independent. $J_n(k)$ is estimated on the basis of 1000 repetitions of the task for $n = 8$. In addition to $\mathbb{E}\,T_2(\bar{U}_n, \bar{V}_n, k)$ the function $\mathbb{E}\,T_4(\bar{U}_n, \bar{V}_n, k)$ is also plotted. The latter can be shown to give $J_n(k)$ correctly up to order $O(n^{-5/2}(k+1)^{-3})$. Notice that although $\mathbb{E}\,T_2(\bar{U}_n, \bar{V}_n, k)$ does not give that good an approximation for $J_n(k)$, its minimizer is near the minimizer of $J_n(k)$, and both of these minimizers lie near the point $\alpha_1/n = 0.125$ as predicted by the theory. In the situation (11) it can actually be shown by lengthy calculations that the minimizer of $J_n(k)$ is exactly $\alpha_1/n$ for each sample size $n \geq 1$.

It is possible to construct cases where $\alpha_1 < 0$. For instance, take

$$X \sim \text{Uniform}(a, b), \quad a = \frac{1}{2}, b = \frac{1}{4}(3\sqrt{5} - 1)$$

$$Y = c/X + d + Z, \quad c = -5, d = 8,$$

and $Z \sim N(0, \sigma^2)$ with $Z$ and $X$ independent and $0 \leq \sigma < 1.1$. In such a case regularization using a positive regularization parameter only makes matters worse; using a properly chosen *negative* regularization parameter would, however, help in this particular case. This would, however, amount to rewarding rapidly changing functions. In the case (11) regularization using a negative value for the regularization parameter would be catastrophic.

## 4   DISCUSSION

We have obtained asymptotic approximations for the optimal regularization parameter in (1) and the amount of improvement (3) in the simple case of one-dimensional linear regression when the regularization parameter is chosen independently of the training sample. It turned out that the optimal regularization parameter is, to leading order, given by $\alpha_1 n^{-1}$ and the resulting improvement is of order $O(n^{-2})$. We have also seen that if $\alpha_1 < 0$ then regularization only makes matters worse.

Also (Larsen and Hansen, 1994) have obtained asymptotic results for the optimal regularization parameter in (1). They consider the case of a nonlinear network; however, they assume that the neural network model is correctly specified.

The generalization of the present results to the nonlinear, misspecified case might be possible using, e.g., techniques from (Bhattacharya and Ghosh, 1978). Generalization to the case where the regularization parameter is chosen on the basis of the sample (say, by cross validation) would be desirable.

**Acknowledgements**

This paper was prepared while the author was visiting the Department for Statistics and Probability Theory at the Vienna University of Technology with financial support from the Academy of Finland. I thank F. Leisch for useful discussions.

## References

Bhattacharya, R. N. and Ghosh, J. K. (1978). On the validity of the formal Edgeworth expansion. *The Annals of Statistics*, 6(2):434–451.

Bishop, C. M. (1995). *Neural Networks for Pattern Recognition*. Oxford University Press.

Brockwell, P. J. and Davis, R. A. (1987). *Time Series: Theory and Methods*. Springer series in statistics. Springer-Verlag.

Hoerl, A. E. and Kennard, R. W. (1988). Ridge regression. In Kotz, S., Johnson, N. L., and Read, C. B., editors, *Encyclopedia of Statistical Sciences*. John Wiley & Sons, Inc.

Larsen, J. and Hansen, L. K. (1994). Generalization performance of regularized neural network models. In Vlontos, J., Whang, J.-N., and Wilson, E., editors, *Proc. of the 4th IEEE Workshop on Neural Networks for Signal Processing*, pages 42–51. IEEE Press.

Ripley, B. D. (1996). *Pattern Recognition and Neural Networks*. Cambridge University Press.

Serfling, R. J. (1980). *Approximation Theorems of Mathematical Statistics*. John Wiley & Sons, Inc.

Titterington, D. M. (1985). Common structure of smoothing techniques in statistics. *International Statistical Review*, 53:141–170.